# $\ell_0$-norm Minimization for Basis Selection

**David Wipf   and   Bhaskar Rao** *

Department of Electrical and Computer Engineering
University of California, San Diego, CA 92092
dwipf@ucsd.edu, brao@ece.ucsd.edu

## Abstract

Finding the sparsest, or minimum $\ell_0$-norm, representation of a signal given an overcomplete dictionary of basis vectors is an important problem in many application domains. Unfortunately, the required optimization problem is often intractable because there is a combinatorial increase in the number of local minima as the number of candidate basis vectors increases. This deficiency has prompted most researchers to instead minimize surrogate measures, such as the $\ell_1$-norm, that lead to more tractable computational methods. The downside of this procedure is that we have now introduced a mismatch between our ultimate goal and our objective function. In this paper, we demonstrate a sparse Bayesian learning-based method of minimizing the $\ell_0$-norm while reducing the number of troublesome local minima. Moreover, we derive necessary conditions for local minima to occur via this approach and empirically demonstrate that there are typically many fewer for general problems of interest.

## 1   Introduction

Sparse signal representations from overcomplete dictionaries find increasing relevance in many application domains [1, 2]. The canonical form of this problem is given by,

$$\min_{\boldsymbol{w}} \|\boldsymbol{w}\|_0, \qquad \text{s.t. } \boldsymbol{t} = \Phi\boldsymbol{w}, \tag{1}$$

where $\Phi \in \Re^{N \times M}$ is a matrix whose columns represent an overcomplete basis (i.e., $\text{rank}(\Phi) = N$ and $M > N$), $\boldsymbol{w}$ is the vector of weights to be learned, and $\boldsymbol{t}$ is the signal vector. The actual cost function being minimized represents the $\ell_0$-norm of $\boldsymbol{w}$ (i.e., a count of the nonzero elements in $\boldsymbol{w}$). In this vein, we seek to find weight vectors whose entries are predominantly zero that nonetheless allow us to accurately represent $\boldsymbol{t}$.

While our objective function is not differentiable, several algorithms have nonetheless been derived that (i), converge almost surely to a solution that locally minimizes (1) and more importantly (ii), when initialized sufficiently close, converge to a maximally sparse solution that also globally optimizes an alternate objective function. For convenience, we will refer these approaches as *local sparsity maximization* (LSM) algorithms. For example, procedures that minimize $\ell_p$-norm-like diversity measures[1] have been developed such that, if $p$ is chosen sufficiently small, we obtain a LSM algorithm [2, 3]. Likewise, a Gaussian entropy-based LSM algorithm called FOCUSS has been developed and successfully employed to

solve Neuromagnetic imaging problems [4]. A similar algorithm was later discovered in [5] from the novel perspective of a Jeffrey's noninformative prior. While all of these methods are potentially very useful candidates for solving (1), they suffer from one significant drawback: as we have discussed in [6], every local minima of (1) is also a local minima to the LSM algorithms.

Unfortunately, there are many local minima to (1). In fact, every basic feasible solution $\boldsymbol{w}_*$ to $\boldsymbol{t} = \Phi\boldsymbol{w}$ is such a local minimum.[2] To see this, we note that the value of $\|\boldsymbol{w}_*\|_0$ at such a solution is less than or equal to $N$. Any other feasible solution can be written as $\boldsymbol{w}_* + \alpha\boldsymbol{w}'$, where $\boldsymbol{w}' \in \text{Null}(\Phi)$. For simplicity, if we assume that every subset of $N$ columns of $\Phi$ are linearly independent, the unique representation property (URP), then $\boldsymbol{w}'$ must necessarily have nonzero elements in locations that differ from $\boldsymbol{w}_*$. Consequently, any solution in the neighborhood of $\boldsymbol{w}_*$ will satisfy $\|\boldsymbol{w}_*\|_0 < \|\boldsymbol{w}_* + \alpha\boldsymbol{w}'\|_0$. This ensures that all such $\boldsymbol{w}_*$ represent local minima to (1).

The number of basic feasible solutions is bounded between $\binom{M-1}{N} + 1$ and $\binom{M}{N}$; the exact number depends on $\boldsymbol{t}$ and $\Phi$ [4]. Regardless, when $M \gg N$, we have an large number of local minima and not surprisingly, we often converge to one of them using currently available LSM algorithms. One potential remedy is to employ a convex surrogate measure in place of the $\ell_0$-norm that leads to a more tractable optimization problem. The most common choice is to use the alternate norm $\|\boldsymbol{w}\|_1$, which creates a unimodal optimization problem that can be solved via linear programming or interior point methods. The considerable price we must pay, however, is that the global minimum of this objective function need not coincide with the sparsest solutions to (1).[3] As such, we may fail to recover the maximally sparse solution regardless of the initialization we use (unlike a LSM procedure).

In this paper, we will demonstrate an alternative algorithm for solving (1) using a sparse Bayesian learning (SBL) framework. Our objective is twofold. First, we will prove that, unlike minimum $\ell_1$-norm methods, the global minimum of the SBL cost function is only achieved at the minimum $\ell_0$-norm solution to $\boldsymbol{t} = \Phi\boldsymbol{w}$. Later, we will show that this method is only locally minimized at a subset of basic feasible solutions and therefore, has fewer local minima than current LSM algorithms.

## 2    Sparse Bayesian Learning

Sparse Bayesian learning was initially developed as a means of performing robust regression using a hierarchal prior that, empirically, has been observed to encourage sparsity [8]. The most basic formulation proceeds as follows. We begin with an assumed likelihood model of our signal $\boldsymbol{t}$ given fixed weights $\boldsymbol{w}$,

$$p(\boldsymbol{t}|\boldsymbol{w}) = (2\pi\sigma^2)^{-N/2} \exp\left(-\frac{1}{2\sigma^2}\|\boldsymbol{t} - \Phi\boldsymbol{w}\|^2\right). \tag{2}$$

To provide a regularizing mechanism, we assume the parameterized weight prior,

$$p(\boldsymbol{w};\boldsymbol{\gamma}) = \prod_{i=1}^{M} (2\pi\gamma_i)^{-1/2} \exp\left(-\frac{w_i^2}{2\gamma_i}\right), \tag{3}$$

where $\boldsymbol{\gamma} = [\gamma_1, \ldots, \gamma_M]^T$ is a vector of $M$ hyperparameters controlling the prior variance of each weight. These hyperparameters (along with the error variance $\sigma^2$ if necessary) can be estimated from the data by marginalizing over the weights and then performing ML optimization. The marginalized pdf is given by

$$p(\boldsymbol{t};\boldsymbol{\gamma}) = \int p(\boldsymbol{t}|\boldsymbol{w})p(\boldsymbol{w};\boldsymbol{\gamma})d\boldsymbol{w} = (2\pi)^{-N/2} |\Sigma_t|^{-1/2} \exp\left[-\frac{1}{2}\boldsymbol{t}^T\Sigma_t^{-1}\boldsymbol{t}\right], \tag{4}$$

where $\Sigma_t \triangleq \sigma^2 I + \Phi\Gamma\Phi^T$ and we have introduced the notation $\Gamma \triangleq \mathrm{diag}(\boldsymbol{\gamma})$.[4] This procedure is referred to as evidence maximization or type-II maximum likelihood [8]. Equivalently, and more conveniently, we may instead *minimize* the cost function

$$\mathcal{L}(\boldsymbol{\gamma}; \sigma^2) = -\log p(\boldsymbol{t}; \boldsymbol{\gamma}) \propto \log|\Sigma_t| + \boldsymbol{t}^T \Sigma_t^{-1} \boldsymbol{t} \tag{5}$$

using the EM algorithm-based update rule for the $(k+1)$-th iteration given by

$$\hat{\boldsymbol{w}}_{(k+1)} \;=\; \mathrm{E}\left[\boldsymbol{w}|\boldsymbol{t}; \boldsymbol{\gamma}_{(k)}\right] = \left(\Phi^T\Phi + \sigma^2\Gamma_{(k)}^{-1}\right)^{-1}\Phi^T\boldsymbol{t} \tag{6}$$

$$\boldsymbol{\gamma}_{(k+1)} \;=\; \mathrm{E}\left[\mathrm{diag}(\boldsymbol{w}\boldsymbol{w}^T)|\boldsymbol{t}; \boldsymbol{\gamma}_{(k)}\right] = \mathrm{diag}\left[\hat{\boldsymbol{w}}_{(k)}\hat{\boldsymbol{w}}_{(k)}^T + \left(\sigma^{-2}\Phi^T\Phi + \Gamma_{(k)}^{-1}\right)^{-1}\right]. \tag{7}$$

Upon convergence to some $\boldsymbol{\gamma}_{ML}$, we compute weight estimates as $\hat{\boldsymbol{w}} = \mathrm{E}[\boldsymbol{w}|\boldsymbol{t}; \boldsymbol{\gamma}_{ML}]$, allowing us to generate $\hat{\boldsymbol{t}} = \Phi\hat{\boldsymbol{w}} \approx \boldsymbol{t}$. We now quantify the relationship between this procedure and $\ell_0$-norm minimization.

## 3  $\ell_0$-norm minimization via SBL

Although SBL was initially developed in a regression context, it can nonetheless be easily adapted to handle (1) by fixing $\sigma^2$ to some $\varepsilon$ and allowing $\varepsilon \to 0$. To accomplish this we must reexpress the SBL iterations to handle the low noise limit. Applying standard matrix identities and the general result

$$\lim_{\varepsilon \to 0} U^T \left(\varepsilon I + UU^T\right)^{-1} = U^\dagger, \tag{8}$$

we arrive at the modified update rules

$$\hat{\boldsymbol{w}}_{(k)} \;=\; \Gamma_{(k)}^{1/2}\left(\Phi\Gamma_{(k)}^{1/2}\right)^\dagger \boldsymbol{t} \tag{9}$$

$$\boldsymbol{\gamma}_{(k+1)} \;=\; \mathrm{diag}\left(\hat{\boldsymbol{w}}_{(k)}\hat{\boldsymbol{w}}_{(k)}^T + \left[I - \Gamma_{(k)}^{1/2}\left(\Phi\Gamma_{(k)}^{1/2}\right)^\dagger \Phi\right]\Gamma_{(k)}\right), \tag{10}$$

where $(\cdot)^\dagger$ denotes the Moore-Penrose pseudo-inverse. We observe that all $\hat{\boldsymbol{w}}_{(k)}$ are feasible, i.e., $\boldsymbol{t} = \Phi\hat{\boldsymbol{w}}_{(k)}$ for all $\boldsymbol{\gamma}_{(k)}$.[5] Also, upon convergence we can easily show that if $\boldsymbol{\gamma}_{ML}$ is sparse, $\hat{\boldsymbol{w}}$ will also be sparse while maintaining feasibility. Thus, we have potentially found an alternative way of solving (1) that is readily computable via the modified iterations above. Perhaps surprisingly, these update rules are equivalent to the Gaussian entropy-based LSM iterations derived in [2, 5], with the exception of the $[I - \Gamma_{(k)}^{1/2}(\Phi\Gamma_{(k)}^{1/2})^\dagger\Phi]\Gamma_{(k)}$ term.

A firm connection with $\ell_0$-norm minimization is realized when we consider the global minimum of $\mathcal{L}(\boldsymbol{\gamma}; \sigma^2 = \varepsilon)$ in the limit as $\varepsilon$ approaches zero. We will now quantify this relationship via the following theorem, which extends results from [6].

**Theorem 1.** Let $\mathcal{W}_0$ denote the set of weight vectors that globally minimize (1). Furthermore, let $\mathcal{W}(\varepsilon)$ be defined as the set of weight vectors

$$\left\{\boldsymbol{w}_{**}: \;\; \boldsymbol{w}_{**} = \left(\Phi^T\Phi + \varepsilon\Gamma_{**}^{-1}\right)^{-1}\Phi^T\boldsymbol{t}, \;\; \boldsymbol{\gamma}_{**} = \arg\min_{\boldsymbol{\gamma}} \mathcal{L}(\boldsymbol{\gamma}; \sigma^2 = \varepsilon)\right\}. \tag{11}$$

Then in the limit as $\varepsilon \to 0$, if $\boldsymbol{w} \in \mathcal{W}(\varepsilon)$, then $\boldsymbol{w} \in \mathcal{W}_0$.

A full proof of this result is available at [9]; however, we provide a brief sketch here. First, we know from [6] that every local minimum of $\mathcal{L}(\boldsymbol{\gamma}; \sigma^2 = \varepsilon)$ is achieved at a basic feasible solution $\boldsymbol{\gamma}_*$ (i.e., a solution with $N$ or fewer nonzero entries), regardless of $\varepsilon$. Therefore, in our search for the global minimum, we only need examine the space of basic feasible solutions. As we allow $\varepsilon$ to become sufficiently small, we show that

$$\mathcal{L}(\boldsymbol{\gamma}_*; \sigma^2 = \varepsilon) = (N - \|\boldsymbol{\gamma}_*\|_0) \log(\varepsilon) + O(1) \tag{12}$$

at any such solution. This result is minimized when $\|\boldsymbol{\gamma}_*\|_0$ is as small as possible. A maximally sparse basic feasible solution, which we denote $\boldsymbol{\gamma}_{**}$, can only occur with nonzero elements aligned with the nonzero elements of some $\boldsymbol{w} \in \mathcal{W}_0$. In the limit as $\varepsilon \to 0$, $\boldsymbol{w}_{**}$ becomes feasible while maintaining the same sparsity profile as $\boldsymbol{\gamma}_{**}$, leading to the stated result.

This result demonstrates that the SBL framework can provide an effective proxy to direct $\ell_0$-norm minimization. More importantly, we will now show that the limiting SBL cost function, which we will henceforth denote

$$\mathcal{L}(\boldsymbol{\gamma}) \triangleq \lim_{\varepsilon \to 0} \mathcal{L}(\boldsymbol{\gamma}; \sigma^2 = \varepsilon) = \log \left| \Phi \Gamma \Phi^T \right| + \boldsymbol{t}^T \left( \Phi \Gamma \Phi^T \right)^{-1} \boldsymbol{t}, \tag{13}$$

need not have the same problematic local minima profile as other methods.

## 4 Analysis of Local Minima

Thus far, we have demonstrated that there is a close affiliation between the limiting SBL framework and the the minimization problem posed by (1). We have not, however, provided any concrete reason why SBL should be preferred over current LSM methods of finding sparse solutions. In fact, this preference is not established until we carefully consider the problem of convergence to local minima.

As already mentioned, the problem with current methods of minimizing $\|\boldsymbol{w}\|_0$ is that every basic feasible solution unavoidably becomes a local minimum. However, what if we could somehow eliminate all or most of these extrema. For example, consider the alternate objective function $f(\boldsymbol{w}) \triangleq \min(\|\boldsymbol{w}\|_0, N)$, leading to the optimization problem

$$\min_{\boldsymbol{w}} f(\boldsymbol{w}), \qquad \text{s.t. } \boldsymbol{t} = \Phi \boldsymbol{w}. \tag{14}$$

While the global minimum remains unchanged, we observe that all local minima occurring at non-degenerate basic feasible solutions have been effectively removed.[6] In other words, at any solution $\boldsymbol{w}_*$ with $N$ nonzero entries, we can always add a small component $\alpha \boldsymbol{w}' \in \text{Null}(\Phi)$ without increasing $f(\boldsymbol{w})$, since $f(\boldsymbol{w})$ can never be greater than $N$. Therefore, we are free to move from basic feasible solution to basic feasible solution without increasing $f(\boldsymbol{w})$. Also, the rare degenerate basic solutions that do remain, even if suboptimal, are sparser by definition. Therefore, locally minimizing our new problem (14) is clearly superior to locally minimizing (1). But how can we implement such a minimization procedure, even approximately, in practice?

Although we cannot remove all non-degenerate local minima and still retain computational feasibility, it is possible to remove many of them, providing some measure of approximation to (14). This is effectively what is accomplished using SBL as will be demonstrated below. Specifically, we will derive necessary conditions required for a non-degenerate basic feasible solution to represent a local minimum to $\mathcal{L}(\boldsymbol{\gamma})$. We will then show that these conditions are frequently *not* satisfied, implying that there are potentially many fewer local minima. Thus, locally minimizing $\mathcal{L}(\boldsymbol{\gamma})$ comes closer to (locally) minimizing (14) than current LSM methods, which in turn, is closer to globally minimizing $\|\boldsymbol{w}\|_0$.

## 4.1 Necessary Conditions for Local Minima

As previously stated, all local minima to $\mathcal{L}(\boldsymbol{\gamma})$ must occur at basic feasible solutions $\boldsymbol{\gamma}_*$. Now suppose that we have found a (non-degenerate) $\boldsymbol{\gamma}_*$ with associated $\boldsymbol{w}_*$ computed via (9) and we would like to assess whether or not it is a local minimum to our SBL cost function. For convenience, let $\widetilde{\boldsymbol{w}}$ denote the $N$ nonzero elements of $\boldsymbol{w}_*$ and $\widetilde{\Phi}$ the associated columns of $\Phi$ (therefore, $\boldsymbol{t} = \widetilde{\Phi}\widetilde{\boldsymbol{w}}$ and $\widetilde{\boldsymbol{w}} = \widetilde{\Phi}^{-1}\boldsymbol{t}$). Intuitively, it would seem likely that if we are not at a true local minimum, then there must exist at least one additional column of $\Phi$ not in $\widetilde{\Phi}$, e.g., some $\boldsymbol{x}$, that is somehow aligned with or in some respect similar to $\boldsymbol{t}$. Moreover, the significance of this potential alignment must be assessed relative to $\widetilde{\Phi}$. But how do we quantify this relationship for the purposes of analyzing local minima?

As it turns out, a useful metric for comparison is realized when we decompose $\boldsymbol{x}$ with respect to $\widetilde{\Phi}$, which forms a basis in $\Re^N$ under the URP assumption. For example, we may form the decomposition $\boldsymbol{x} = \widetilde{\Phi}\widetilde{\boldsymbol{v}}$, where $\widetilde{\boldsymbol{v}}$ is a vector of weights analogous to $\widetilde{\boldsymbol{w}}$. As will be shown below, the similarity required between $\boldsymbol{x}$ and $\boldsymbol{t}$ (needed for establishing the existence of a local minimum) may then be realized by comparing the respective weights $\widetilde{\boldsymbol{v}}$ and $\widetilde{\boldsymbol{w}}$. In more familiar terms, this is analogous to suggesting that similar signals have similar Fourier expansions. Loosely, we may expect that if $\widetilde{\boldsymbol{v}}$ is 'close enough' to $\widetilde{\boldsymbol{w}}$, then $\boldsymbol{x}$ is sufficiently close to $\boldsymbol{t}$ (relative to all other columns in $\widetilde{\Phi}$) such that we are not at a local minimum. We formalize this idea via the following theorem:

**Theorem 2.** Let $\Phi$ satisfy the URP and let $\boldsymbol{\gamma}_*$ represent a vector of hyperparameters with $N$ and only $N$ nonzero entries and associated basic feasible solution $\widetilde{\boldsymbol{w}} = \widetilde{\Phi}^{-1}\boldsymbol{t}$. Let $\mathcal{X}$ denote the set of $M - N$ columns of $\Phi$ not included in $\widetilde{\Phi}$ and $\mathcal{V}$ the set of weights given by $\left\{\widetilde{\boldsymbol{v}} : \widetilde{\boldsymbol{v}} = \widetilde{\Phi}^{-1}\boldsymbol{x}, \boldsymbol{x} \in \mathcal{X}\right\}$. Then $\boldsymbol{\gamma}_*$ is a local minimum of $\mathcal{L}(\boldsymbol{\gamma})$ only if

$$\sum_{i \neq j} \frac{\widetilde{v}_i \widetilde{v}_j}{\widetilde{w}_i \widetilde{w}_j} < 0 \qquad \forall \widetilde{\boldsymbol{v}} \in \mathcal{V}. \tag{15}$$

*Proof*: If $\boldsymbol{\gamma}_*$ truly represents a local minimum of our cost function, then the following condition must hold for all $\boldsymbol{x} \in \mathcal{X}$:

$$\frac{\partial \mathcal{L}(\boldsymbol{\gamma}_*)}{\partial \gamma_x} \geq 0, \tag{16}$$

where $\gamma_x$ denotes the hyperparameter corresponding to the basis vector $\boldsymbol{x}$. In words, we cannot reduce $\mathcal{L}(\boldsymbol{\gamma}_*)$ along a positive gradient because this would push $\gamma_x$ below zero. Using the matrix inversion lemma, the determinant identity, and some algebraic manipulations, we arrive at the expression

$$\frac{\partial \mathcal{L}(\boldsymbol{\gamma}_*)}{\partial \gamma_x} = \frac{\boldsymbol{x}^T B \boldsymbol{x}}{1 + \gamma_x \boldsymbol{x}^T B \boldsymbol{x}} - \left(\frac{\boldsymbol{t}^T B \boldsymbol{x}}{1 + \gamma_x \boldsymbol{x}^T B \boldsymbol{x}}\right)^2, \tag{17}$$

where $B \triangleq (\widetilde{\Phi}\widetilde{\Gamma}\widetilde{\Phi}^T)^{-1}$. Since we have assumed that we are at a local minimum, it is straightforward to show that $\widetilde{\Gamma} = \text{diag}(\widetilde{\boldsymbol{w}})^2$ leading to the expression

$$B = \widetilde{\Phi}^{-T} \text{diag}(\widetilde{\boldsymbol{w}})^{-2} \widetilde{\Phi}^{-1}. \tag{18}$$

Substituting this expression into (17) and evaluating at the point $\gamma_x = 0$, the above gradient reduces to

$$\frac{\partial \mathcal{L}(\boldsymbol{\gamma}_*)}{\partial \gamma_x} = \widetilde{\boldsymbol{v}}^T \left(\text{diag}(\widetilde{\boldsymbol{w}}^{-1}\widetilde{\boldsymbol{w}}^{-T}) - \widetilde{\boldsymbol{w}}^{-1}\widetilde{\boldsymbol{w}}^{-T}\right)\widetilde{\boldsymbol{v}}, \tag{19}$$

where $\widetilde{\boldsymbol{w}}^{-1} \triangleq [\widetilde{w}_1^{-1}, \ldots, \widetilde{w}_N^{-1}]^T$. This leads directly to the stated theorem. ∎

This theorem provides a useful picture of what is required for local minima to exist and more importantly, why many basic feasible solutions are not local minimum. Moreover, there are several convenient ways in which we can interpret this result to accommodate a more intuitive perspective.

## 4.2 A Simple Geometric Interpretation

In general terms, if the signs of each of the elements in a given $\widetilde{v}$ match up with $\widetilde{w}$, then the specified condition will be violated and we cannot be at a local minimum. We can illustrate this geometrically as follows.

To begin, we note that our cost function $\mathcal{L}(\gamma)$ is invariant with respect to reflections of any basis vectors about the origin, i.e., we can multiply any column of $\Phi$ by $-1$ and the cost function does not change. Returning to a candidate local minimum with associated $\widetilde{\Phi}$, we may therefore assume, without loss of generality, that $\widetilde{\Phi} \equiv \widetilde{\Phi}\text{diag}\left(\text{sgn}(w)\right)$, giving us the decomposition $t = \widetilde{\Phi}w$, $w > 0$. Under this assumption, we see that $t$ is located in the convex cone formed by the columns of $\widetilde{\Phi}$. We can infer that if any $x \in \mathcal{X}$ (i.e., any column of $\Phi$ not in $\widetilde{\Phi}$) lies in this convex cone, then the associated coefficients $\widetilde{v}$ must all be positive by definition (likewise, by a similar argument, any $x$ in the convex cone of $-\widetilde{\Phi}$ leads to the same result). Consequently, Theorem 2 ensures that we are not at a local minimum. The simple 2D example shown in Figure 1 helps to illustrate this point.

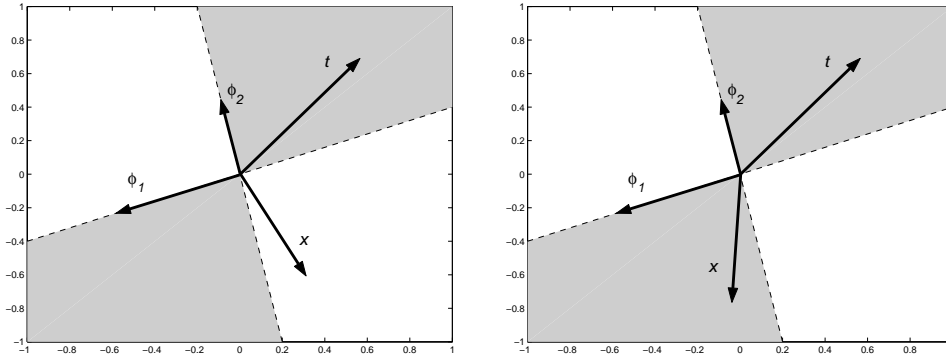

Figure 1: 2D example with a $2 \times 3$ dictionary $\Phi$ (i.e., $N = 2$ and $M = 3$) and a basic feasible solution using the columns $\widetilde{\Phi} = [\phi_1 \ \phi_2]$. *Left*: In this case, $x = \phi_3$ does not penetrate the convex cone containing $t$, and we do not satisfy the conditions of Theorem 2. This configuration does represent a minimizing basic feasible solution. *Right*: Now $x$ is in the cone and therefore, we know that we are not at a local minimum; but this configuration *does* represent a local minimum to current LSM methods.

Alternatively, we can cast this geometric perspective in terms of relative cone sizes. For example, let $C_{\widetilde{\Phi}}$ represent the convex cone (and its reflection) formed by $\widetilde{\Phi}$. Then we are not at a local minimum to $L(\gamma)$ if there exists a second convex cone $C$ formed from a subset of columns of $\Phi$ such that $t \in C \subset C_{\widetilde{\Phi}}$, i.e., $C$ is a tighter cone containing $t$. In Figure 1(*right*), we obtain a tighter cone by swapping $x$ for $\phi_2$.

While certainly useful, we must emphasize that in higher dimensions, these geometric conditions are *much* weaker than (15), e.g., if all $x$ are *not* in the convex cone of $\widetilde{\Phi}$, we still may not be at a local minimum. In fact, to guarantee a local minimum, all $x$ must be reasonably far from this cone as quantified by (15). Of course the ultimate reduction in local minima from the $\binom{M-1}{N} + 1$ to $\binom{M}{N}$ bounds is dependent on the distribution of

| M/N | 1.3 | 1.6 | 2.0 | 2.4 | 3.0 |
|---|---|---|---|---|---|
| SBL Local Minimum % | 4.9% | 4.0% | 3.2% | 2.3% | 1.6% |

Table 1: Given 1000 trials where FOCUSS has converged to a suboptimal local minimum, we tabulate the percentage of times the local minimum is also a local minimum to SBL. $M/N$ refers to the overcompleteness ratio of the dictionary used, with $N$ fixed at 20. Results using other algorithms are similar.

basis vectors in $t$-space. In general, it is difficult to quantify this reduction except in a few special cases.[7] However, we will now proceed to empirically demonstrate that the overall reduction in local minima is substantial when the basis vectors are randomly distributed.

## 5 Empirical Comparisons

To show that the potential reduction in local minima derived above translates into concrete results, we conducted a simulation study using randomized basis vectors distributed isometrically in $t$-space. Randomized dictionaries are of interest in signal processing and other disciplines [2, 7] and represent a viable benchmark for testing basis selection methods. Moreover, we have performed analogous experiments with other dictionary types (such as pairs of orthobases) leading to similar results (see [9] for some examples).

Our goal was to demonstrate that current LSM algorithms often converge to local minima that do not exist in the SBL cost function. To accomplish this, we repeated the following procedure for dictionaries of various sizes. First, we generate a random $N \times M$ $\Phi$ whose columns are each drawn uniformly from a unit sphere. Sparse weight vectors $w_0$ are randomly generated with $\|w_0\|_0 = 7$ (and uniformly distributed amplitudes on the nonzero components). The vector of target values is then computed as $t = \Phi w_0$. The LSM algorithm is then presented with $t$ and $\Phi$ and attempts to learn the minimum $\ell_0$-norm solutions. The experiment is repeated a sufficient number of times such that we collect 1000 examples where the LSM algorithm converges to a local minimum. In all these cases, we check if the condition stipulated by Theorem 2 applies, allowing us to determine if the given solution is a local minimum to the SBL algorithm or not. The results are contained in Table 1 for the FOCUSS LSM algorithm. We note that, the larger the overcompleteness ratio $M/N$, the larger the total number of LSM local minima (via the bounds presented earlier). However, there also appears to be a greater probability that SBL can avoid any given one.

In many cases where we found that SBL was not locally minimized, we initialized the SBL algorithm in this location and observed whether or not it converged to the optimal solution. In roughly 50% of these cases, *it escaped to find the maximally sparse solution.* The remaining times, it did escape in accordance with theory; however, it converged to another local minimum. In contrast, when we initialize other LSM algorithms at an SBL local minima, we always remain trapped as expected.

## 6 Discussion

In practice, we have consistently observed that SBL outperforms current LSM algorithms in finding maximally sparse solutions (e.g., see [9]). The results of this paper provide a very plausible explanation for this improved performance: conventional LSM procedures are very likely to converge to local minima that do not exist in the SBL landscape. However,

it may still be unclear exactly why this happens. In conclusion, we give a brief explanation that provides insight into this issue.

Consider the canonical FOCUSS LSM algorithm or the Figueiredo algorithm from [5] (with $\sigma^2$ fixed to zero, the Figueiredo algorithm is actually equivalent to the FOCUSS algorithm). These methods essentially solve the problem

$$\min_{\boldsymbol{w}} \sum_{i=1}^{M} \log |w_i|, \qquad \text{s.t. } \boldsymbol{t} = \Phi \boldsymbol{w}, \tag{20}$$

where the objective function is proportional to the Gaussian entropy measure. In contrast, we can show that, up to a scale factor, any minimum of $\mathcal{L}(\boldsymbol{\gamma})$ must also be a minimum of

$$\min_{\boldsymbol{\gamma}} \sum_{i=1}^{N} \log \lambda_i(\boldsymbol{\gamma}), \qquad \text{s.t. } \boldsymbol{\gamma} \in \Omega_{\boldsymbol{\gamma}}, \tag{21}$$

where $\lambda_i(\boldsymbol{\gamma})$ is the $i$-th eigenvalue of $\Phi \Gamma \Phi^T$ and $\Omega_{\boldsymbol{\gamma}}$ is the convex set $\{\boldsymbol{\gamma} : \boldsymbol{t}^T \left( \Phi \Gamma \Phi^T \right)^{-1} \boldsymbol{t} \leq 1, \boldsymbol{\gamma} \geq 0\}$.

In both instances, we are minimizing a Gaussian entropy measure over a convex constraint set. The crucial difference resides in the particular parameterization applied to this measure. In (20), we see that if *any* subset of $|w_i|$'s becomes significantly small (e.g., as we approach a basic feasible solution), we enter the basin of a local minimum because the associated $\log |w_i|$ terms becomes enormously negative; hence the one-to-one correspondence between basic feasible solutions and local minima of the LSM algorithms.

In contrast, when working with (21), many of the $\gamma_i$'s may approach zero without becoming trapped, as long as $\Phi \Gamma \Phi^T$ remains reasonably well-conditioned. In other words, since $\Phi$ is overcomplete, up to $M - N$ of the $\gamma_i$'s can be zero while still maintaining a full set of nonzero eigenvalues to $\Phi \Gamma \Phi^T$, so no term in the summation is driven towards minus infinity as occurred above. Thus, we can switch from one basic feasible solution to another in many instances while still reducing our objective function. It is in this respect that SBL approximates the minimization of the alternative objective posed by (14).

## Footnotes

*This work was supported by an ARCS Foundation scholarship, DiMI grant 22-8376 and Nissan.

[1]Minimizing a diversity measure is often equivalent to maximizing sparsity.

[2]A basic feasible solution is a solution with at most $N$ nonzero entries.

[3]In very restrictive settings, it has been shown that the minimum $\ell_1$-norm solution can equal the minimum $\ell_0$-norm solution [7]. But in practical situations, this result often does not apply.

[4] We will sometimes use $\Gamma$ and $\boldsymbol{\gamma}$ interchangeably when appropriate.

[5] This assumes that $\boldsymbol{t}$ is in the span of the columns of $\Phi$ associated with nonzero elements in $\boldsymbol{\gamma}$, which will always be the case if $\boldsymbol{t}$ is in the span of $\Phi$ and all $\boldsymbol{\gamma}$ are initialized to nonzero values.

[6]A *degenerate* basic feasible solution has strictly less than $N$ nonzero entries; however, the vast majority of local minima are non-degenerate, containing exactly $N$ nonzero entries.

[7]For example, in the special case where $t$ is proportional to a single column of $\Phi$, we can show that the number of local minima reduces from $\binom{M-1}{N}+1$ to 1, i.e., we are left with a single minimum.

## References

[1] S.S. Chen, D.L. Donoho, and M.A. Saunders, "Atomic decomposition by basis pursuit," *SIAM Journal on Scientific Computing*, vol. 20, no. 1, pp. 33–61, 1999.

[2] B.D. Rao and K. Kreutz-Delgado, "An affine scaling methodology for best basis selection," *IEEE Transactions on Signal Processing*, vol. 47, no. 1, pp. 187–200, January 1999.

[3] R.M. Leahy and B.D. Jeffs, "On the design of maximally sparse beamforming arrays," *IEEE Transactions on Antennas and Propagation*, vol. 39, no. 8, pp. 1178–1187, Aug. 1991.

[4] I. F. Gorodnitsky and B. D. Rao, "Sparse signal reconstruction from limited data using FOCUSS: A re-weighted minimum norm algorithm," *IEEE Transactions on Signal Processing*, vol. 45, no. 3, pp. 600–616, March 1997.

[5] M.A.T. Figueiredo, "Adaptive sparseness using Jeffreys prior," *Neural Information Processing Systems*, vol. 14, pp. 697–704, 2002.

[6] D.P. Wipf and B.D. Rao, "Sparse Bayesian learning for basis selection," *IEEE Transactions on Signal Processing*, vol. 52, no. 8, pp. 2153–2164, 2004.

[7] D.L. Donoho and M. Elad, "Optimally sparse representation in general (nonorthogonal) dictionaries via $\ell_1$ minimization," *Proc. National Academy of Sciences*, vol. 100, no. 5, pp. 2197–2202, March 2003.

[8] M.E. Tipping, "Sparse Bayesian learning and the relevance vector machine," *Journal of Machine Learning Research*, vol. 1, pp. 211–244, 2001.

[9] D.P. Wipf and B.D. Rao, "Some results on sparse Bayesian learning," *ECE Department Technical Report*, University of California, San Diego, 2005.
